# Wormholes Improve Contrastive Divergence

**Geoffrey Hinton, Max Welling and Andriy Mnih**
Department of Computer Science, University of Toronto
10 King's College Road, Toronto, M5S 3G5 Canada
{*hinton,welling,amnih*}*@cs.toronto.edu*

## Abstract

In models that define probabilities via energies, maximum likelihood learning typically involves using Markov Chain Monte Carlo to sample from the model's distribution. If the Markov chain is started at the data distribution, learning often works well even if the chain is only run for a few time steps [3]. But if the data distribution contains modes separated by regions of very low density, brief MCMC will not ensure that different modes have the correct relative energies because it cannot move particles from one mode to another. We show how to improve brief MCMC by allowing long-range moves that are suggested by the data distribution. If the model is approximately correct, these long-range moves have a reasonable acceptance rate.

## 1 Introduction

One way to model the density of high-dimensional data is to use a set of parameters, $\Theta$ to deterministically assign an energy, $E(\mathbf{x}|\Theta)$ to each possible datavector, $\mathbf{x}$ [2].

$$p(\mathbf{x}|\Theta) = \frac{e^{-E(\mathbf{x}|\Theta)}}{\int e^{-E(\mathbf{y}|\Theta)}d\mathbf{y}} \qquad (1)$$

The obvious way to fit such an energy-based model to a set of training data is to follow the gradient of the likelihood. The contribution of a training case, $\mathbf{x}$, to the gradient is:

$$\frac{\partial \log p(\mathbf{x}|\Theta)}{\partial \theta_j} = -\frac{\partial E(\mathbf{x}|\Theta)}{\partial \theta_j} + \int p(\mathbf{y}|\Theta)\frac{\partial E(\mathbf{y}|\Theta)}{\partial \theta_j}\,d\mathbf{y} \qquad (2)$$

The last term in equation 2 is an integral over all possible datavectors and is usually intractable, but it can be approximated by running a Markov chain to get samples from the Boltzmann distribution defined by the model's current parameters. The main problem with this approach is the time that it takes for the Markov chain to approach its stationary distribution. Fortunately, in [3] it was shown that if the chain is started at the data distribution, running the chain for just a few steps is often sufficient to provide a signal for learning. The way in which the data distribution gets distorted by the model in the first few steps of the Markov chain provides enough information about how the model differs from reality to allow the parameters of the model to be improved by lowering the energy of the data and raising the energy of the "confabulations" produced by a few steps of the Markov chain. So the steepest ascent learning algorithm implied by equation 2 becomes

$$\Delta\theta_j \propto -\left\langle\frac{\partial E(.|\Theta)}{\partial \theta_j}\right\rangle_{data} + \left\langle\frac{\partial E(.|\Theta)}{\partial \theta_j}\right\rangle_{confabulations} \qquad (3)$$

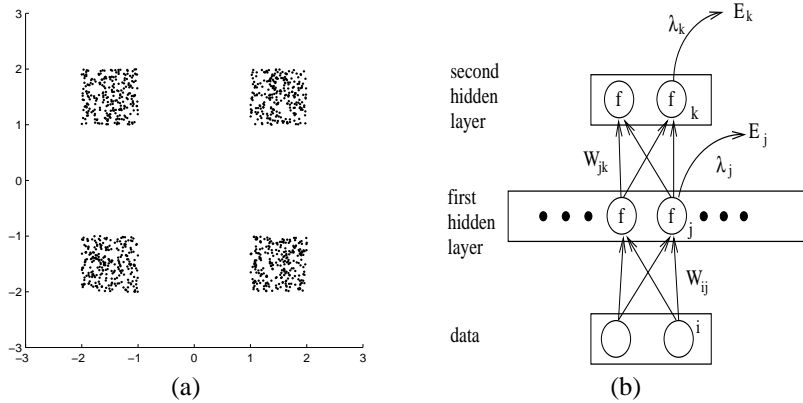

Figure 1: a) shows a two-dimensional data distribution that has four well-separated modes. b) shows a feedforward neural network that is used to assign an energy to a two-dimensional input vector. Each hidden unit takes a weighted sum of its inputs, adds a learned bias, and puts this sum through a logistic non-linearity to produce an output that is sent to the next layer. Each hidden unit makes a contribution to the global energy that is equal to its output times a learned scale factor. There are 20 units in the first hidden layer and 3 in the top layer.

where the angle brackets denote expected values under the distribution specified by the subscript.

If we use a Markov chain that obeys detailed balance, it is clear that when the training data is dense and the model is perfect, the learning procedure in equation 3 will leave the parameters unchanged because the Markov chain will already be at its stationary distribution so the confabulations will have the same distribution as the training data.

Unfortunately, real training sets may have modes that are separated by regions of very low density, and running the Markov chain for only a few steps may not allow it to move between these modes even when there is a lot of data. As a result, the relative energies of data points in different modes can be completely wrong without affecting the learning signal given by equation 3. The point of this paper is to show that, in the context of model-fitting, there are ways to use the known training data to introduce extra mode-hopping moves into the Markov chain. We rely on the observation that after some initial training, the training data itself provides useful suggestions about where the modes of the model are and how much probability mass there is in each mode.

## 2   A simple example of wormholes

Figure 1a shows some two-dimensional training data and a model that was used to model the density of the training data. The model is an unsupervised deterministic feedforward neural network with two hidden layers of logistic units. The parameters of the model are the weights and biases of the hidden units and one additional scale parameter per hidden unit which is used to convert the output of the hidden unit into an additive contribution to the global energy. By using backpropagation through the model, it is easy to compute the derivatives of the global energy assigned to an input vector *w.r.t.* the parameters (needed in equation 3), and it is also easy to compute the gradient of the energy *w.r.t.* each component of the input vector (*i.e* the slope of the energy surface at that point in dataspace). The latter gradient is needed for the 'Hybrid Monte Carlo' sampler that we discuss next.

The model is trained on 1024 datapoints for 1000 parameter updates using equation 3. To produce the confabulations we start at the datapoints and use a Markov chain that is a sim-

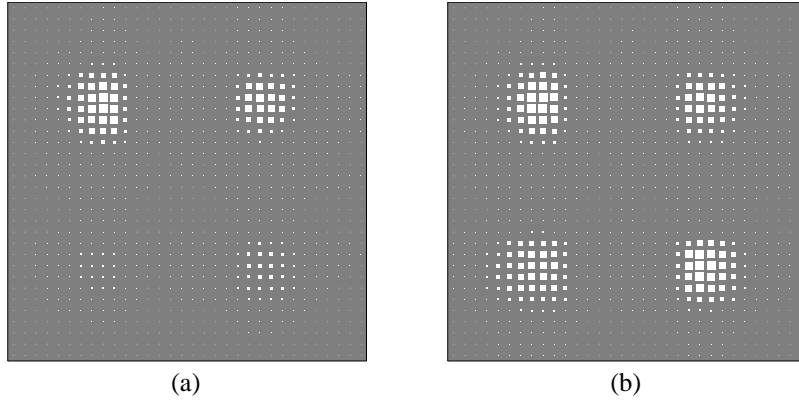

<center>(a)                                     (b)</center>

Figure 2: (a) shows the probabilities learned by the network without using wormholes, displayed on a $32 \times 32$ grid in the dataspace. Some modes have much too little probability mass. (b) shows that the probability mass in the different minima matches the data distribution after 10 parameter updates using point-to-point wormholes defined by the vector differences between pairs of training points. The mode-hopping allowed by the wormholes increases the number of confabulations that end up in the deeper minima which causes the learning algorithm to raise the energy of these minima.

plified version of Hybrid Monte Carlo. Each datapoint is treated as a particle on the energy surface. The particle is given a random initial momentum chosen from a unit-variance isotropic Gaussian and its deterministic trajectory along the energy surface is then simulated for 10 time steps. If this simulation has no numerical errors the increase, $\Delta E$, in the combined potential and kinetic energy will be zero. If $\Delta E$ is positive, the particle is returned to its initial position with a probability of $1 - exp(-\Delta E)$. The step size is adapted after each batch of trajectories so that only about $10\%$ of the trajectories get rejected. Numerical errors up to second order are eliminated by using a "leapfrog" method [5] which uses the potential energy gradient at time $t$ to compute the velocity increment between time $t - \frac{1}{2}$ and $t + \frac{1}{2}$ and uses the velocity at time $t + \frac{1}{2}$ to compute the position increment between time $t$ and $t + 1$.

Figure 2a shows the probability density over the two-dimensional space. Notice that the model assigns much more probability mass to some minima than to others. It is clear that the learning procedure in equation 3 would correct this imbalance if the confabulations were generated by a time-consuming Markov chain that was able to concentrate the confabulations in the deepest minima[1], but we want to make use of the data distribution to achieve the same goal much faster.

Figure 2b shows how the probability density is corrected by 10 parameter updates using a Markov chain that has been modified by adding an optional long-range jump at the end of each accepted trajectory. The candidate jump is simply the vector difference between two randomly selected training points. The jump is always accepted if it lowers the energy. If it raises the energy it is accepted with a probability of $exp(-\Delta E)$. Since the probability that point A in the space will be offered a jump to point B is the same as the probability that B will be offered a jump to A, the jumps do not affect detailed balance. One way to think about the jumps is to imagine that every point in the dataspace is connected by wormholes to $n(n-1)$ other points so that it can move to any of these points in a single step.

To understand how the long-range moves deal with the trade-off between energy and entropy, consider a proposed move that is based on the vector offset between a training point

that lies in a deep narrow energy minimum and a training point that lies in a broad shallow minimum. If the move is applied to a random point in the deep minimum, it stands a good chance of moving to a point within the broad shallow minimum, but it will probably be rejected because the energy has increased. If the opposite move is applied to a random point in the broad minimum, the resulting point is unlikely to fall within the narrow minimum, though if it does it is very likely to be accepted. If the two minima have the same free energy, these two effects exactly balance.

Jumps generated by random pairs of datapoints work well if the minima are all the same shape, but in a high-dimensional space it is very unlikely that such a jump will be accepted if different energy minima are strongly elongated in different directions.

## 3    A local optimization-based method

In high dimensions the simple wormhole method will have a low acceptance rate because most jumps will land in high-energy regions. One way avoid to that is to use local optimization: after a jump has been made descend into a nearby low-energy region. The obvious difficulty with this approach is that care must be taken to preserve detailed balance. We use a variation on the method proposed in [7]. It fits Gaussians to the detected low energy regions in order to account for their volume.

A Gaussian is fitted using the following procedure. Given a point $x$, let $m_x$ be the point found by running a minimization algorithm on $E(x)$ for a few steps (or until convergence) starting at $x$. Let $H_x$ be the Hessian of $E(x)$ at $m_x$, adjusted to ensure that it is positive definite by adding a multiple of the identity matrix to it. Let $\Sigma_x$ be the inverse of $H_x$. A Gaussian density $g_x(y)$ is then defined by the mean $m_x$ and the covariance matrix $\Sigma_x$.

To generate a jump proposal, we make a forward jump by adding the vector difference $d$ between two randomly selected data points to the initial point $x_0$, obtaining $x$. Then we compute $m_x$ and $\Sigma_x$, and sample a proposed jump destination $y$ from $g_x(y)$. Then we make a backward jump by adding $-d$ to $y$ to obtain $z$, and compute $m_z$ and $\Sigma_z$, specifying $g_z(x)$. Finally, we accept the proposal $y$ with probability

$$p = \min(1, \frac{\exp(-E(y))}{\exp(-E(x_0))} \frac{g_z(x_0)}{g_x(y)}).$$

Our implementation of the algorithm executes 20 steps of steepest descent to find $m_x$ and $m_z$. To save time, instead of computing the full Hessian, we compute a diagonal approximation to the Hessian using the method proposed in [1].

## 4    Gaping wormholes

In this section we describe a third method based on "darting MCMC" [8] to jump between the modes of a distribution. The idea of this technique is to define spherical regions on the modes of the distribution and to jump only between corresponding points in those regions. When we consider a long-range move we check whether or not we are inside a wormhole. When inside a wormhole we initiate a jump to some other wormhole (e.g. chosen uniformly); when outside we stay put in order to maintain detailed balance. If we make a jump we must also use the usual Metropolis rejection rule to decide whether to accept the jump.

In high dimensional spaces this procedure may still lead to unacceptably high rejection rates because the modes will likely decay sharply in at least a few directions. Since these ridges of probability are likely to be uncorrelated across the modes, the proposed target location of the jump will most of the time have very low probability, resulting in almost

certain rejection. To deal with this problem, we propose a generalization of the described method, where the wormholes can have arbitrary shapes and volumes. As before, when we are considering a long-range move we first check our position, and if we are located inside a wormhole we initiate a jump (which may be rejected) while if we are located outside a wormhole we stay put. To maintain detailed balance between wormholes we need to compensate for their potentially different volume factors. To that end, we impose the constraint

$$V_i \, P_{i \rightarrow j} = V_j \, P_{j \rightarrow i} \tag{4}$$

on all pairs of wormholes, where $P_{i \rightarrow j}$ is a transition probability and $V_i$ and $V_j$ are the volumes of the wormholes $i$ and $j$ respectively. This in fact defines a separate Markov chain between the wormholes with equilibrium distribution,

$$P_i^{\mathbf{EQ}} = \frac{V_i}{\sum_j V_j} \tag{5}$$

The simplest method[2] to compensate for the different volume factors is therefore to sample a target wormhole from this distribution $P^{\mathbf{EQ}}$. When the target wormhole has been determined we can either sample a point uniformly within its volume or design some deterministic mapping (see also [4]). Finally, once the arrival point has been determined we need to compensate for the fact that the probability of the point of departure is likely to be different than the probability of the point of arrival. The usual Metropolis rule applies in this case,

$$P_{accept} = \min \left[ 1, \frac{P_{arrive}}{P_{depart}} \right] \tag{6}$$

This combined set of rules ensures that detailed balance holds and that the samples will eventually come from the correct probability distribution. One way of employing this sampler in conjunction with contrastive divergence learning is to fit a "mixture of Gaussians" model to the data distribution in a preprocessing step. The region inside an iso-probability contour of each Gaussian mixture component defines an elliptical wormhole with volume

$$V_{ellipse} = \frac{\pi^{\frac{d}{2}} \alpha^d \prod_{i=1}^d \sigma_i}{\Gamma(1 + \frac{d}{2})} \tag{7}$$

where $\Gamma(x)$ is the gamma function, $\sigma_i$ is the standard deviation of the i'$^{th}$ eigen-direction of the covariance matrix and $\alpha$ is a free parameter controlling the size of the wormhole. These regions provide good jump points during CD-learning because it is expected that the valleys in the energy landscape correspond to the regions where the data cluster. To minimize the rejection rate we map points in one ellipse to "corresponding" points in another ellipse as follows. Let $\Sigma_{depart}$ and $\Sigma_{arrive}$ be the covariance matrices of the wormholes in question. and let $\Sigma = USU^T$ be an eigenvalue decomposition. The following transformation maps iso-probability contours in one wormhole to iso-probability contours in another,

$$x_{arrive} - \mu_{arrive} = -U_{arrive} S_{arrive}^{1/2} S_{depart}^{-1/2} U_{depart}^T (x_{depart} - \mu_{depart}) \tag{8}$$

with $\mu$ the center location of the ellipse. The negative sign in front of the transformation is to promote better exploration when the target wormhole turns out to be the same as the wormhole from which the jump is initiated. It is important to realize that although the mapping is one-to-one, we still need to satisfy the constraint in equation 4 because a volume element $dx$ will change under the mapping. Thus, wormholes are sampled from $P^{\mathbf{EQ}}$ and proposed moves are accepted according to equation 6.

For both the deterministic and the stochastic moves we may also want to consider regions that overlap. For instance, if we generate wormholes by fitting a mixture of Gaussians it

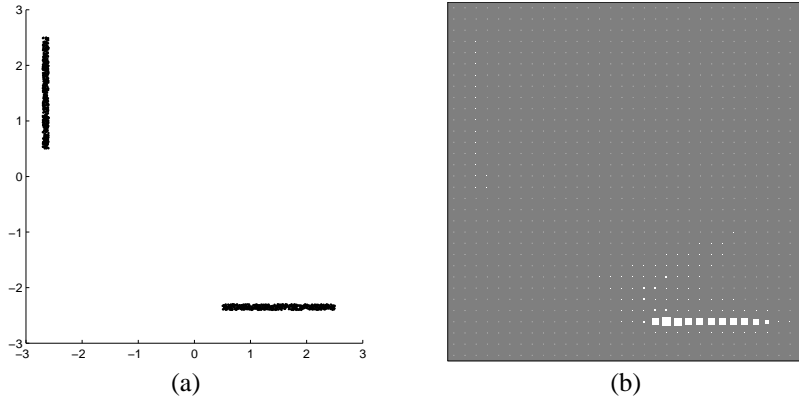

(a)                                           (b)

Figure 3: (a) Dataset of 1024 cases uniformly distributed on 2 orthogonal narrow rectangles. (b) Probability density of the model learned with contrastive divergence. The size of each square indicates the probability mass at the corresponding location.

is very hard to check whether these regions overlap somewhere in space. Fortunately, we can adapt the sampling procedure to deal with this case as well. First define $n_{arrive}$ as the total number of regions that contain point $x_{arrive}$ and similarly for $n_{depart}$. Detailed balance can still be maintained for both deterministic and stochastic moves if we adapt the Metropolis acceptance rule as follows,

$$P_{accept} = \min \left[ 1, \frac{n_{depart} P_{arrive}}{n_{arrive} P_{depart}} \right] \qquad (9)$$

Further details can be found in [6].

## 5  An experimental comparison of the three methods

To highlight the difference between the point and the region wormhole sampler, we sampled 1024 data points along two very narrow orthogonal ridges (see figure 3a), with half of the cases in each mode. A model with the same architecture as depicted in figure 1 was learned using contrastive divergence, but with "Cauchy" nonlinearities of the form $f(x) = \log(1 + x^2)$ instead of the logistic function. The probability density of the model that resulted is shown in figure 3b. Clearly, the lack of mixing between the modes has resulted in one mode being much stronger than the other one. Subsequently, learning was resumed using a Markov chain that proposed a long-range jump for all confabulations after each brief HMC run. The regions in the region wormhole sampler were generated by fitting a mixture of two Gaussians to the data using EM, and setting $\alpha = 10$. Both the point wormhole method and the region wormhole method were able to correct the asymmetry in the solution but the region method does so much faster as shown in figure 4b. The reason is that a much smaller fraction of the confabulations succeed in making a long-range jump as shown in figure 4a.

We then compared all three wormhole algorithms on a family of datasets of varying dimensionality. Each dataset contained 1024 $n$-dimensional points, where $n$ was one of 2, 4, 8, 16, or 32. The first two components of each point were sampled uniformly from two axis-aligned narrow orthogonal ridges and then rotated by $45°$ around the origin to ensure that the diagonal approximation to the Hessian, used by the local optimization-based algorithm, was not unfairly accurate. The remaining $n - 2$ components of each data point were sampled independently from a sharp univariate Gaussian with mean 0 and *std.* 0.02.

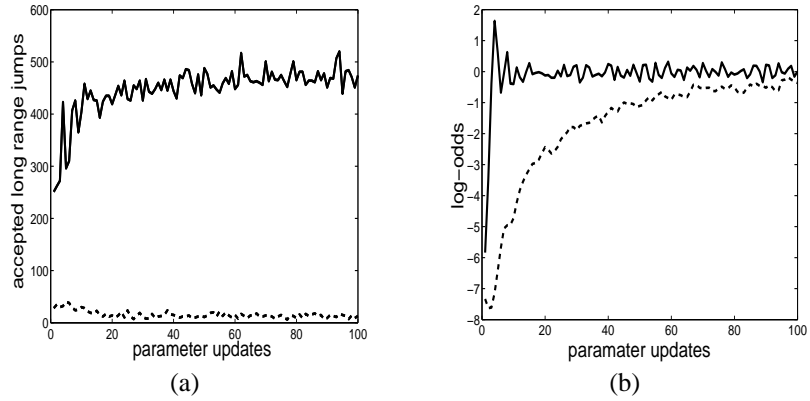

<center>(a)                                                    (b)</center>

Figure 4: (a) Number of successful jumps between the modes for point wormhole MCMC (dashed line) and region wormhole MCMC (solid line). (b) Log-odds of the probability masses contained in small volumes surrounding the two modes for the point wormhole method (dashed line) and the region wormhole method (solid line). The log-odds is zero when the probability mass is equal in both modes.

The networks used for comparison had architectures identical to the one depicted in Figure 1 in all respects except for the number and the type of units used. The second hidden layer consisted of Cauchy units, while the first hidden layer consisted of some Cauchy and some sigmoid units. The networks were trained for 2000 parameter updates using HMC without wormholes. To speed up the training, an adaptive learning rate and a momentum of 0.95 were used. We also used a weight decay rate of 0.0001 for weights and 0.000001 for scales. Gaussian noise was added to the last $n-2$ components of each data point. The *std.* of the noise started at $0.2$ and was gradually decreased to zero as training progressed. This prevented HMC from being slowed down by the narrow energy ravines resulting from the tight constraints on the last $n-2$ components.

After the model was trained (without wormholes), we compared the performance of the three jump samplers by allowing each sampler to make a proposal for each training case and then comparing the acceptance rates. This was repeated 25 times to improve the estimate of the acceptance rate. In each sampler, HMC was run for 10 steps before offering points an opportunity to jump.

The average number of successful jumps between modes per iteration is shown in the table below.

| Dimensionality | Network architecture | Simple wormholes | Optimization-based | Region wormholes |
|---|---|---|---|---|
| 2 | 10+10, 2 | 10 | 15 | 372 |
| 4 | 20+10, 4 | 6 | 17 | 407 |
| 8 | 20+10, 6 | 3 | 19 | 397 |
| 16 | 40+10, 8 | 1 | 13 | 338 |
| 32 | 50+10, 10 | 1 | 9 | 295 |
|  | Relative run time | 1 | 2.6 | 1 |

The network architecture column shows the number of units in the hidden layers with each entry giving the number of Cauchy units plus the number of sigmoid units in the first hidden layer and the number of Cauchy units in the second hidden layer.

# 6 Summary

Maximum likelihood learning of energy-based models is hard because the gradient of the log probability of the data with respect to the parameters depends on the distribution defined by the model and it is computationally expensive to even get samples from this distribution. Minimizing contrastive divergence is much easier than maximizing likelihood but the brief Markov chain does not have time to mix between separated modes in the distribution[3]. The result is that the local structure around each data cluster is modelled well, but the relative masses of different cluster are not. In this paper we proposed three algorithms to deal with this phenomenon. Their success relies on the fact that the data distribution provides valuable suggestions about the location of the modes of a good model. Since the probability of the model distribution is expected to be substantial in these regions they can be successfully used as target locations for long-range moves in a MCMC sampler.

The MCMC sampler with point-to-point wormholes is simple but has a high rejection rate when the modes are not aligned. Performing local gradient descent after a jump significantly increases the acceptance rate, but only leads to a modest improvement in efficiency because of the extra computations required to maintain detailed balance. The MCMC sampler with region-to-region wormholes targets its moves to regions that are likely to have high probability under the model and therefore has a much better acceptance rate, provided the distribution can be modelled well by a mixture. None of the methods we have proposed will work well for high-dimensional, approximately factorial distributions that have exponentially many modes formed by the cross-product of multiple lower-dimensional distributions.

**Acknowledgements** This research was funded by NSERC, CFI, OIT. We thank Radford Neal and Yee-Whye Teh for helpful advice and Sam Roweis for providing software.

## Footnotes

[1]Note that depending on the height of the energy barrier between the modes this may take too long for practical purposes.

[2]Other methods that respect the constraint 4 are possible but are suboptimal in the sense that they mix slower to the equilibrium distribution.

[3]However, note that in cases where the modes are well separated, even Markov chains that run for an extraordinarily long time will not mix properly between those modes, and the results of this paper become relevant.

# References

[1] S. Becker and Y. LeCun. Improving the convergence of back-propagation learning with sec ond-order methods. In D. Touretzky, G. Hinton, and T. Sejnowski, editors, *Proc. of the 1988 Connectionist Models Summer School*, pages 29–37, San Mateo, 1989. Morgan Kaufman.

[2] Y. Bengio, R. Ducharme, and P. Vincent. A neural probabilistic language model. In *Advances in Neural Information Processing Systems, 2001*, 2001.

[3] G.E. Hinton. Training products of experts by minimizing contrastive divergence. *Neural Computation*, 14:1771–1800, 2002.

[4] C. Jarzynski. Targeted free energy perturbation. Technical Report LAUR-01-2157, Los Alamos National Laboratory, 2001.

[5] R.M. Neal. Probabilistic inference using markov chain monte carlo methods. Technical Report CRG-TR-93-1, University of Toronto, Computer Science, 1993.

[6] C. Sminchisescu, M.Welling, and G. Hinton. Generalized darting monte carlo. Technical report, University of Toronto, 2003. Technical Report CSRG-478.

[7] H. Tjelemeland and B.K. Hegstad. Mode jumping proposals in mcmc. Technical report, Norwegian University of Science and Technology, Trondheim, Norway, 1999. Rep. No. Statistics no.1/1999.

[8] A. Voter. A monte carlo method for determining free-energy differences and transition state theory rate constants. 82(4), 1985.

